# The power of feature clustering: An application to object detection

**Shai Avidan**
Mitsubishi Electric Research Labs
201 Broadway
Cambridge, MA 02139
*avidan@merl.com*

**Moshe Butman**
Adyoron Intelligent Systems LTD.
34 Habarzel St.
Tel-Aviv, Israel
*mosheb@adyoron.com*

## Abstract

We give a fast rejection scheme that is based on image segments and demonstrate it on the canonical example of face detection. However, instead of focusing on the detection step we focus on the rejection step and show that our method is simple and fast to be learned, thus making it an excellent pre-processing step to accelerate standard machine learning classifiers, such as neural-networks, Bayes classifiers or SVM. We decompose a collection of face images into regions of pixels with similar behavior over the image set. The relationships between the mean and variance of image segments are used to form a cascade of rejectors that can reject over $99.8\%$ of image patches, thus only a small fraction of the image patches must be passed to a full-scale classifier. Moreover, the training time for our method is much less than an hour, on a standard PC. The shape of the features (i.e. image segments) we use is data-driven, they are very cheap to compute and they form a very low dimensional feature space in which exhaustive search for the best features is tractable.

## 1 Introduction

This work is motivated by recent advances in object detection algorithms that use a cascade of rejectors to quickly detect objects in images. Instead of using a full fledged classifier on every image patch, a sequence of increasingly more complex rejectors is applied. Non-face image patches will be rejected early on in the cascade, while face image patches will survive the entire cascade and will be marked as a face.

The work of Viola & Jones [15] demonstrated the advantages of such an approach. Other researchers suggested similar methods [4, 6, 12]. Common to all these methods is the realization that simple and fast classifiers are enough to reject large portions of the image, leaving more time to use more sophisticated, and time consuming, classifiers on the remaining regions of the image.

All these "fast" methods must address three issues. First, is the feature space in which to work, second is a fast method to calculate the features from the raw image data and third is the feature selection algorithm to use.

Early attempts assumed the feature space to be the space of pixel values. Elad *et al.* [4]

suggest the maximum rejection criteria that chooses rejectors that maximize the rejection rate of each classifier. Keren *et al.* [6] use *anti-face* detectors by assuming normal distribution on the background. A different approach was suggested by Romdhani *et al.* [12], that constructed the full SVM classifier first and then approximated it with a sequence or support vector rejectors that were calculated using non-linear optimization. All the above mentioned method need to "touch" every pixel in an image patch at least once before they can reject the image patch.

Viola & Jones [15], on the other hand, construct a huge feature space that consists of combined box regions that can be quickly computed from the raw pixel data using the "integral image" and use a sequential feature selection algorithm for feature selection. The rejectors are combined using a variant of AdaBoost [2]. Li *et al* [7] replaced the sequential forward searching algorithm with a float search algorithm (which can backtrack as well). An important advantage of the huge feature space advocated by Viola & Jones is that now image patches can be rejected with an extremely small number of operations and there is no need to "touch" every pixel in the image patch at least once.

Many of these methods focus on developing fast classifiers that are often constructed in a greedy manner. This precludes classifiers that might demonstrate excellent classification results but are slower to compute, such as the methods suggested by Schneiderman *et al.* [8], Rowley *et al.* [13], Sung and Poggio [10] or Heisele *et al* [5].

Our method offers a way to accelerate "slow" classification methods by using a pre-processing rejection step. Our rejection scheme is fast to be trained and very effective in rejecting the vast majority of false patterns. On the canonical face detection example, it took our method much less than an hour to train and it was able to reject over $99.8\%$ of the image patches, meaning that we can effectively accelerate standard classifiers by several orders of magnitude, without changing the classifier at all.

Like other, "fast", methods we use a cascade of rejectors, but we use a different type of filters and a different type of feature selection method. We take our features to be the approximated mean and variance of image segments, where every image segment consists of pixels that have similar behavior across the entire image set. As a result, our features are derived from the data and do not have to be hand crafted for the particular object of interest. In fact they do not even have to form contiguous regions. We use only a small number of representative pixels to calculate the approximated mean and variance, which makes our features very fast to compute during detection (in our experiments we found that our first rejector rejects almost $50\%$ of all image patches, using just $8$ pixels). Finally, the number of segments we use is quite small which makes it possible to exhaustively calculate all possible rejectors based on single, pairs and triplets of segments in order to find the best rejectors in every step of the cascade. This is in contrast to methods that construct a huge feature bank and use a greedy feature selection algorithm to choose "good" features from it. Taken together, our algorithm is fast to train and fast to test. In our experiments we train on a database that contains several thousands of face images and roughly half-a-million non-faces in less than an hour on an average PC and our rejection module runs at several frames per second.

## 2 Algorithm

At the core of our algorithm is the realization that feature representation is a crucial ingredient in any classification system. For instance, the Viola-Jones box filters are extremely efficient to compute using the "integral image" but they form a large feature space, thus placing a heavy computational burden on the feature selection algorithm that follows. Moreover, empirically they show that the first feature selected by their method correspond to meaningful regions in the face. This suggests that it might be better to focus on features that

correspond to coherent regions in the image. This leads to the idea of image segmentation, that breaks an ensemble of images into regions of pixels that exhibit similar temporal behavior. Given the image segmentation we take our features to be the mean and variance of each segment, giving us a very small feature space to work on (we chose to segment the face image into eight segments). Unfortunately, calculating the mean and variance of an image segment requires going over all the pixels in the segment, a time consuming process. However, since the segments represent similar-behaving pixels we found that we can approximate the calculation of the mean and variance of the entire segment using quite a small number of representative pixels. In our experiments, four pixels were enough to adequately represent segments that contain several tens of pixels. Now that we have a very small feature space to work with, and a fast way to extract features from raw pixels data we can exhaustively search for all possible combinations of single, pairs or triplets of features to find the best rejector in every stage. The remaining patterns should be passed to a standard classifier for final validation.

## 2.1 Image Segments

Image segments were already presented in the past [1] for the problem of classification of objects such as faces or vehicles. We briefly repeat the presentation for the paper to be self-contained. An ensemble of scaled, cropped and aligned images of a given object (say faces) can be approximated by its leading principal components. This is done by stacking the images (in vector form) in a design matrix $\mathbf{A}$ and taking the leading eigenvectors of the covariance matrix $\mathbf{C} = \frac{1}{N}\mathbf{A}\mathbf{A}^T$, where $N$ is the number of images. The leading principal components are the leading eigenvectors of the covariance matrix $\mathbf{C}$ and they form a basis that approximates the space of all the columns of the design matrix $\mathbf{A}$ [11, 9]. But instead of looking at the *columns* of $\mathbf{A}$ look at the *rows* of $\mathbf{A}$. Each row in $\mathbf{A}$ gives the intensity profile of a particular pixel, i.e., each row represents the intensity values that a particular pixel takes in the different images in the ensemble. If two pixels come from the same region of the face they are likely to have the same intensity values and hence have a strong temporal correlation. We wish to find this correlations and segment the image plane into regions of pixels that have similar temporal behavior. This approach broadly falls under the category of Factor Analysis [3] that seeks to find a low-dimensional representation that captures the correlations between features.

Let $\mathbf{A}^x$ be the $x$-th *row* of the design matrix $\mathbf{A}$. Then $\mathbf{A}^x$ is the intensity profile of pixel $x$ (We address pixels with a single number because the images are represented in a scan-line vector form). That is, $\mathbf{A}^x$ is an $N$-dimensional vector (where $N$ is the number of images) that holds the intensity values of pixel $x$ in each image in the ensemble. Pixels $x$ and $y$ are temporally correlated if the dot product of rows $\mathbf{A}^x$ and $\mathbf{A}^y$ is approaching 1 and are temporally uncorrelated if the dot-product is approaching 0.

Thus, to find temporally correlated pixels all we need to do is run a clustering algorithm on the rows of the design matrix $\mathbf{A}$. In particular, we used the k-means algorithm on the rows of the matrix $\mathbf{A}$ but any method of Factor Analysis can be used. As a result, the image-plane is segmented into several (possibly non-continuous) segments of temporally correlated pixels. Experiments in the past [1] showed good classification results on different objects such as faces and vehicles.

## 2.2 Finding Representative Pixels

Our algorithm works by comparing the mean and variance properties of one or more image segments. Unfortunately this requires touching every pixel in the image segment during test time, thus slowing the classification process considerably. Therefor, during train time we find a set of representative pixels that will be used during test time. Specifically, we approximate every segment in a face image with a small number of representative pixels

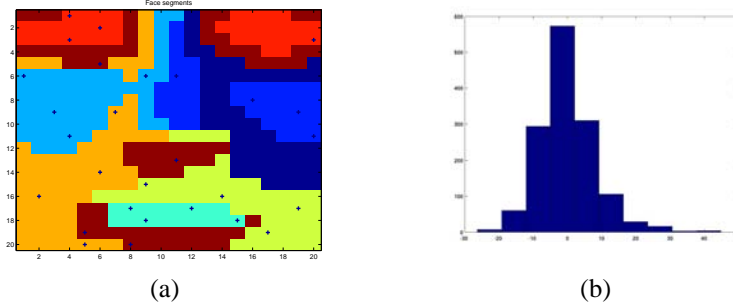

(a)                                                      (b)

Figure 1: Face segmentation and representative pixels. (a) Face segmentation and representative pixels. The face segmentation was computed using 1400 faces, each segment is marked with a different color and the segments need not be contiguous. The crosses overlaid on the segments mark the representative pixels that were automatically selected by our method. (b) Histogram of the difference between an approximated mean and the exact mean of a particular segment (the light blue segment on the left). The histogram is peaked at zero, meaning that the representative pixels give a good approximation.

that approximate the mean and variance of the entire image segment. Define $\mu_i(\mathbf{x_j})$ to be the true mean of segment $i$ of face $j$, and let $\hat{\mu}_i(\mathbf{x_j})$ be its approximation, defined as

$$\hat{\mu}_i(\mathbf{x_j}) = \frac{\sum_{j=1}^{k} x_j}{k}$$

where $\{x_j\}_{j=1}^{k}$ are a subset of pixels in segment $i$ of pattern $j$. We use a greedy algorithm that incrementally searches for the next representative pixel that minimize

$$\sum_{j=1}^{n} (\hat{\mu}_i(\mathbf{x_j})) - \mu_i(\mathbf{x_j}))^2$$

and add it to the collection of representative pixels of segment $i$. In practice we use four representative pixels per segment. The representative pixels computed this way are used for computing both the approximated mean and the approximated variance of every test pattern. Figure 1 show how well this approximation works in practice.

Given the representative pixels, the approximated variance $\hat{\sigma}_i(\mathbf{x_j})$ of segment $i$ of pattern $j$ is given by:

$$\hat{\sigma}_i(\mathbf{x_j}) = \sum_{j=1}^{k} |x_j - \hat{\mu}_i(\mathbf{x_j})|$$

## 2.3 The rejection cascade

We construct a rejection cascade that can quickly reject image patches, with minimal computational load. Our feature space consist of the approximated mean and variance of the image segments. In our experiments we have 8 segments, each represented by its mean and variance, giving rise to a $16D$ feature space. This feature space is very fast to compute, as we need only four pixels to calculate the approximate mean and variance of the segment. Because the feature space is so small we can exhaustively search for all classifiers on single, pairs and triplets of segments. In addition this feature space gives enough information to reject texture-less regions without the need to normalize the mean or variance of the entire image patch. We next describe our rejectors in detail.

### 2.3.1 Feature rejectors

Now, that we have segmented every image into several segments and approximated every segment with a small number of representative pixels, we can exhaustively search for the best combination of segments that will reject the largest number of non-face images. We repeat this process until the improvement in rejection is negligible.

Given a training set of $P$ positive examples (i.e. faces) and $N$ negative examples we construct the following linear rejectors and adjust the parameter $\theta$ so that they will correctly classify $d \cdot P$ (we use $d = 0.95$) of the face images and save $r$, the number of negative examples they correctly rejected, as well as the parameter $\theta$.

1. For each segment $i$, find a bound on its approximated mean. Formally, find $\theta$ s.t.
$$\hat{\mu}_i(\mathbf{x}) > \theta \ \ or \ \ \hat{\mu}_i(\mathbf{x}) < \theta$$

2. For each segment $i$, find a bound on its approximated variance. Formally, find $\theta$ s.t.
$$\hat{\sigma}_i(\mathbf{x}) > \theta \ \ or \ \ \hat{\sigma}_i(\mathbf{x}) < \theta$$

3. For each pair of segments $i, j$, find a bound on the difference between their approximated means. Formally, find $\theta$ s.t.
$$\hat{\mu}_i(\mathbf{x}) - \hat{\mu}_j(\mathbf{x}) > \theta \ \ or \ \ \hat{\mu}_i(\mathbf{x}) - \hat{\mu}_j(\mathbf{x}) < \theta$$

4. For each pair of segments $i, j$, find a bound on the difference between their approximated variance. Formally, find $\theta$ s.t.
$$\hat{\sigma}_i(\mathbf{x}) - \hat{\sigma}_j(\mathbf{x}) > \theta \ \ or \ \ \hat{\sigma}_i(\mathbf{x}) - \hat{\sigma}_j(\mathbf{x}) < \theta$$

5. For each triplet of segments $i, j, k$ find a bound on the difference of the absolute difference of their approximated means. Formally, find $\theta$ s.t.
$$|\hat{\mu}_i(\mathbf{x}) - \hat{\mu}_j(\mathbf{x})| - |\hat{\mu}_i(\mathbf{x}) - \hat{\mu}_k(\mathbf{x})| > \theta$$

This process is done only once to form a pool of rejectors. We do not re-train rejectors after selecting a particular rejector.

### 2.3.2 Training

We form the cascade of rejectors from a large pattern vs. rejector binary table $\mathbf{T}$, where each entry $\mathbf{T}(i, j)$ is 1 if rejector $j$ rejects pattern $i$. Because the table is binary we can store every entry in a single bit and therefor a table of $513,000$ patterns and $664$ rejectors can easily fit in the memory. We then use a greedy algorithm to pick the next rejector with the highest rejection score $r$. We repeat this process until $r$ falls below some predefined threshold.

1. Sum each column and choose column (rejector) $j$ with the highest sum.
2. For each entry $T(i, j)$, in column $j$, that is equal to one, zero row $i$.
3. Go to step 1

The entire process is extremely fast and takes only several minutes, including I/O. The idea of creating a rejector pool in advance was independently suggested by [16] to accelerate the Viola-Jones training time. We obtain 50 rejectors using this method. Figure 2a shows the rejection rate of this cascade on a training set of $513,000$ images, as well as the number of arithmetic operations it takes. Note that roughly $50\%$ of all patterns are rejected by the first rejector using only 12 operations. During testing we compute the approximated mean and variance only when they are needed and not before hand.

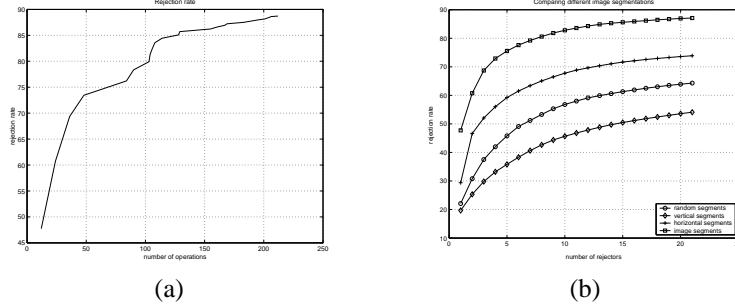

|  |  |
|---|---|
| (a) | (b) |

Figure 2: (a) Rejection rate on training set. The x-axis counts the number of arithmetic operations needed for rejection. The y-axis is the rejection rate on a training set of about half-a-million non-faces and about $1500$ faces. Note that almost $50\%$ of the false patterns are rejected with just 12 operations. Overall rejection rate of the feature rejectors on the training set is $88\%$, it drops to about $80\%$ on the CMU+MIT database. (b) Rejection rate as a function of image segmentation method. We trained our system using four types of image segmentation and show the rejector. We compare our image segmentation approach against naive segmentation of the image plane into horizontal blocks, vertical blocks or random segmentation. In each case we trained a cascade of 21 rejectors and calculated their accumulative rejection rate on our training set. Clearly working with our image segments gives the best results.

We wanted to confirm our intuition that indeed only meaningful regions in the image can produce such results and we therefor performed the following experiment. We segmented the pixels in the image using four different methods. (1) using our image segments (2) into 8 horizontal blocks (3) into 8 vertical blocks (4) into 8 randomly generated segments. Figure 2b show that image segments gives the best results, by far.

The remaining false positive patterns are passed on to the next rejectors, as described next.

### 2.4 Texture-less region rejection

We found that the feature rejectors defined in the previous section are doing poorly in rejecting texture-less regions. This is because we do not perform any sort of variance normalization on the image patch, a step that will slow us down. However, by now we have computed the approximated mean and variance of all the image segments and we can construct rejectors based on all of them to reject texture-less regions. In particular we construct the following two rejectors

1. Reject all image patches where the variance of all $8$ approximated means falls below a threshold. Formally, find $\theta$ s.t.

$$\hat{\sigma}(\hat{\mu}_i(\mathbf{x})) < \theta \quad i = 1...8$$

2. Reject all image patches where the variance of all $8$ approximated variances falls below a threshold. Formally, find $\theta$ s.t.

$$\hat{\sigma}(\hat{\sigma}_i(\mathbf{x})) < \theta \quad i = 1...8$$

### 2.5 Linear classifier

Finally, we construct a cascade of 10 linear rejectors, using all 16 features (i.e. the approximated means and variance of all 8 segments).

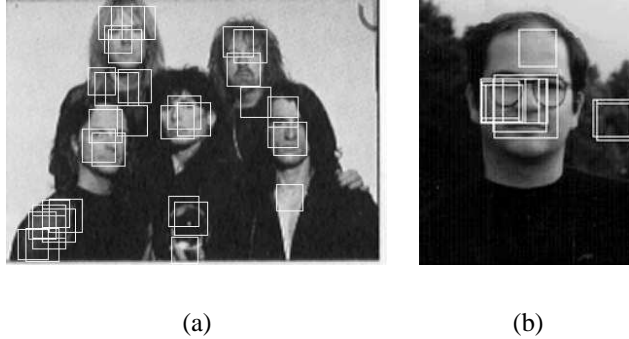

<div align="center">(a)                     (b)</div>

Figure 3: Examples. We show examples from the CMU+MIT dataset. Our method correctly rejected over $99.8\%$ of the image patches in the image, leaving only a handful of image patches to be tested by a "slow", full scale classifier.

### 2.6 Multi-detection heuristic

As noted by previous authors [15] face classifiers are insensitive to small changes in position and scale and therefor we adopt the heuristic that only four overlapping detections are declared a face. This help reduce the number of detected rectangles around and face, as well as reject some spurious false detections.

## 3 Experiments

We have tested our rejection scheme on the standard CMU+MIT database [13]. We created a pyramid at increasing scales of $1.1$ and scanned every scale for rectangles of size $20 \times 20$ in jumps of two pixels. We calculate the approximated mean and variance only when they are needed, to save time.

Overall, our rejection scheme rejected over $99.8\%$ of the image patches, while correctly detecting $93\%$ of the faces. On average the feature rejectors rejected roughly $80\%$ of all image patches, the textureless region rejectors rejected additional $10\%$ of the image patches, the linear rejectors rejected additional $5\%$ and the multi-detection heuristic rejected the remaining image patterns. The average rejection rate per image is over $99.8\%$. This is not enough for face detection, as there are roughly $615,000$ image patches per image in the CMU+MIT database, and our rejector cascade passes, on average, $870$ false positive image patches, per image. This patterns will have to be passed to a full-scale classifier to be properly rejected. Figure 3 give some examples of our system. Note that the system correctly detects all the faces, while allowing a small number of false positives.

We have also experimented with rescaling the features, instead of rescaling the image, but noted that the number of false positives increased by about $5\%$ for every fixed detection rate we tried (All the results reported here use image pyramids).

## 4 Summary and Conclusions

We presented a fast rejection scheme that is based on image segments and demonstrated it on the canonical example of face detection. Image segments are made of regions of pixels with similar behavior over the image set. The shape of the features (i.e. image segments) we use is data-driven and they are very cheap to compute The relationships between the mean and variance of image segments are used to form a cascade of rejectors that can reject over $99.8\%$ of the image patches, thus only a small fraction of the image patches must be

passed to a full-scale classifier. The training time for our method is much less than an hour, on a standard PC. We believe that our method can be used to accelerate standard machine learning algorithms that are too slow for object detection, by serving as a gate keeper that rejects most of the false patterns.

## References

[1] Shai Avidan. EigenSegments: A spatio-temporal decomposition of an ensemble of image. In European Conference on Computer Vision (ECCV), May 2002, Copenhagen, Denmark.

[2] Yoav Freund and Robert E. Schapire. A decision-theoretic generalization of on-line learning and an application to boosting. In Computational Learning Theory: Eurocolt 95, pages 2337. Springer-Verlag, 1995.

[3] R. O. Duda and P. E. Hart. Pattern Classification and Scene Analysis. Wiley-Interscience publication, 1973.

[4] M. Elad, Y. Hel-Or and R. Keshet. Rejection based classifier for face detection. Pattern Recognition Letters 23 (2002) 1459-1471.

[5] B. Heisele, T. Serre, S. Mukherjee, and T. Poggio. Feature reduction and hierarchy of classifiers for fast object detection in video images. In Proc. CVPR, volume 2, pages 1824, 2001.

[6] D. Keren, M. Osadchy, and C. Gotsman. Antifaces: A novel, fast method for image detection. IEEE Trans. on Pattern Analysis and Machine Intelligence, 23(7):747761, 2001.

[7] S.Z. Li, L. Zhu, Z.Q. Zhang, A. Blake, H.J. Zhang and H. Shum. Statistical Learning of Multi-View Face Detection. In *Proceedings of the 7th European Conference on Computer Vision*, Copenhagen, Denmark, May 2002.

[8] Henry Schneiderman and Takeo Kanade. A statistical model for 3d object detection applied to faces and cars. In IEEE Conference on Computer Vision and Pattern Recognition. IEEE, June 2000.

[9] L. Sirovich and M. Kirby. Low-dimensional procedure for the characterization of human faces. In *Journal of the Optical Society of America 4*, 510-524.

[10] K.-K. Sung and T. Poggio. Example-based Learning for View-Based Human Face Detection. In *IEEE Transactions on Pattern Analysis and Machine Intelligence* 20(1):39-51, 1998.

[11] M. Turk and A. Pentland. Eigenfaces for recognition. In *Journal of Cognitive Neuroscience*, vol. 3, no. 1, 1991.

[12] S. Romdhani, P. Torr, B. Schoelkopf, and A. Blake. Computationally efficient face detection. In Proc. Intl. Conf. Computer Vision, pages 695700, 2001.

[13] H. A. Rowley, S. Baluja, and T. Kanade. Neural network-based face detection. IEEE Trans. on Pattern Analysis and Machine Intelligence, 20(1):2338, 1998.

[14] V. Vapnik. *The Nature of Statistical Learning Theory.* Springer, N.Y., 1995.

[15] P. Viola and M. Jones. Rapid Object Detection using a Boosted Cascade of Simple Features. In *IEEE Conference on Computer Vision and Pattern Recognition*, Hawaii, 2001.

[16] J. Wu, J. M. Rehg, and M. D. Mullin. Learning a Rare Event Detection Cascade by Direct Feature Selection. To appear in Advances in Neural Information Processing Systems 16 (NIPS*2003), MIT Press, 2004.
